# A Note on the Representational Incompatibility of Function Approximation and Factored Dynamics

**Eric Allender**
Computer Science Department
Rutgers University
allender@cs.rutgers.edu

**Sanjeev Arora**
Computer Science Department
Princeton University
arora@cs.princeton.edu

**Michael Kearns**
Department of Computer and Information Science
University of Pennsylvania
mkearns@cis.upenn.edu

**Cristopher Moore**
Department of Computer Science
University of New Mexico
moore@santafe.edu

**Alexander Russell**
Department of Computer Science and Engineering
University of Connecticut
acr@cse.uconn.edu

## Abstract

We establish a new hardness result that shows that the difficulty of planning in factored Markov decision processes is *representational* rather than just computational. More precisely, we give a *fixed* family of factored MDPs with linear rewards whose optimal policies and value functions simply cannot be represented succinctly in *any* standard parametric form. Previous hardness results indicated that *computing* good policies from the MDP parameters was difficult, but left open the possibility of succinct function approximation for any fixed factored MDP. Our result applies even to policies which yield a polynomially poor approximation to the optimal value, and highlights interesting connections with the complexity class of *Arthur-Merlin games*.

## 1 Introduction

While a number of different representational approaches to large Markov decision processes (MDPs) have been proposed and studied over recent years, relatively little is known about the relationships between them. For example, in function approximation, a parametric form is proposed for the value functions of policies. Presumably, for any assumed parametric form (for instance, linear value functions), rather strong constraints on the underlying stochastic dynamics and rewards may be required to meet the assumption. However, a precise characterization of such constraints seems elusive.

Similarly, there has been recent interest in making parametric assumptions on the dynamics and rewards directly, as in the recent work on factored MDPs. Here it is known that the problem of computing an optimal policy from the MDP parameters is intractable (see [7] and the references therein), but exactly what the representational constraints on such policies are has remained largely unexplored.

In this note, we give a new intractability result for planning in factored MDPs that exposes a noteworthy conceptual point missing from previous hardness results. Prior intractability results for planning in factored MDPs established that the problem of *computing* optimal policies from MDP parameters is hard, but left open the possibility that for any *fixed* factored MDP, there might exist a compact, parametric representation of its optimal policy. This would be roughly analogous to standard NP-complete problems such as graph coloring — any 3-colorable graph has a "compact" description of its 3-coloring, but it is hard to compute it from the graph.

Here we dismiss even this possibility. Under a standard and widely believed complexity-theoretic assumption (that is even weaker than the assumption that NP does not have polynomial size Boolean circuits), we prove that a specific family of factored MDPs does not even *possess* "succinct" policies. By this we mean something extremely general — namely, that for each MDP in the family, it cannot have an optimal policy represented by an arbitrary Boolean circuit whose size is bounded by a polynomial in the size of the MDP description. Since such circuits can represent essentially any standard parametric functional form, we are showing that there exists no "reasonable" representation of good policies in factored MDPs, even if we ignore the problem of how to compute them from the MDP description. This result holds even if we ask only for policies whose expected return *approximates* the optimal within a polynomial factor. (With a slightly stronger complexity-theoretic assumption, it follows that obtaining an approximation even within an *exponential* factor is impossible.)

Thus, while previous results established that there was at least a *computational* barrier to going from factored MDP parameters to good policies, here we show that the barrier is actually *representational*, a considerably worse situation. The result highlights the fact that even when making strong and reasonable assumptions about one representational aspect of MDPs (such as value functions or dynamics), there is no reason in general for this to lead to any nontrivial restrictions on the others.

The construction in our result is ultimately rather simple, and relies on powerful results developed in complexity theory over the last decade. In particular, we exploit striking results on the complexity class associated with computational protocols known as *Arthur-Merlin games*.

We note that recent and independent work by Liberatore [5] establishes results similar to ours. The primary differences between our work and Liberatore's is that our results prove intractability of approximation and rely on different proof techniques.

## 2   DBN-Markov Decision Processes

A *Markov decision process* is a tuple $(Q, A, \delta, r)$ where $Q$ is a set of *states*, $A$ is a set of *actions*, $\delta = \langle \delta_{q,a} \rangle$ is a family of probability distributions on $Q$, one for each $(q, a) \in Q \times A$, and $r : Q \to \mathbb{R}^+$ is a reward function. We will denote by $\delta_{q,a}(q')$ the probability that action $a$ in state $q$ results in state $q'$. When started in a state $q_0$, and provided with a sequence of actions $a_0, a_1, \ldots$, the MDP traverses a sequence of states $q_0, q_1, q_2, \ldots$, where each $q_{i+1}$ is a random sample from the distribution $\delta_{q_i,a_i}(\cdot)$. Such a state sequence is called a *path*. The $\gamma$-*discounted return* associated with such a path is $r_\gamma(q_0, q_1, \ldots) = \sum_{i=1}^{\infty} r(q_i)\gamma^i$.

A *policy* $\pi : Q \to A$ is a mapping from states to actions. When the action sequence is generated according to this policy, we denote by $\mathbf{p}[q_0; \pi] = (q_0; q_1, \ldots)$ the state sequence produced as above. A policy $\pi$ is *optimal* if for all policies $\pi'$ and all $q \in Q$, we have $\mathrm{Exp}[r_\gamma(\mathbf{p}[q, \pi])] \geq \mathrm{Exp}[r_\gamma(\mathbf{p}[q, \pi'])]$.

We consider MDPs where the transition law $\delta$ is represented as a *dynamic Bayes net*, or DBN-MDPs. Namely, if the state space $Q$ has size $2^n$, then $\delta$ is represented by a 2-layer Bayes net. There are $n + 1$ variables in the first layer, representing the $n$ state variables at any given time $t$, along with the action chosen at time $t$. There are $n$ variables in the second layer, representing the $n$ state variables at time $t + 1$. All directed edges in the Bayes net go from variables in the first layer to variables in the second layer; for our result, it suffices to consider Bayes nets in which the indegree of every second-layer node is bounded by some constant. Each second layer node has a conditional probability table (CPT) describing its conditional distribution for every possible setting of its parents in the Bayes net. Thus the stochastic dynamics of the DBN-MDP are entirely described by the Bayes net in the standard way; the next-state distribution for any state is given by simply fixing the first layer nodes to the settings given by the state. Any given action choice then yields the next-state distribution according to standard Bayes net semantics. We shall assume throughout that the rewards are a linear function of state.

## 3    Arthur-Merlin Games

The complexity class AM is a probabilistic extension of the familiar class NP, and is typically described in terms of *Arthur–Merlin* games (see [2]). An Arthur–Merlin game for a language $L$ is played by two players (Turing machines), $V$ (the Verifier, often referred to as *Arthur* in the literature), who is equipped with a random coin and only modest (polynomial-time bounded) computing power; and $P$ (the Prover, often referred to as *Merlin*), who is computationally unbounded. Both are supplied with the same input $x$ of length $n$ bits. For instance, $x$ might be some standard encoding of an undirected graph $G$, and $P$ might be interested in proving to $V$ that $G$ is 3-colorable. Thus, $P$ seeks to prove that $x \in L$; $V$ is skeptical but willing to listen. At each step of the conversation, $V$ flips a fair coin, perhaps several times, and reports the resulting bits to $V$; this is interpreted as a "question" or "challenge" to $P$. In the graph coloring example, it might be reasonable to interpret the random bits generated by $V$ as identifying a random edge in $G$, with the challenge to $P$ being to identify the colors of the nodes on each end of this edge (which had better be different, and consistent with any previous responses of $P$, if $V$ is to be convinced). Thus $P$ responds with some number of bits, and the protocol proceeds to the next round. After $\mathrm{poly}(n)$ steps, $V$ decides, based upon the conversation, whether to *accept* that $x \in L$ or *reject*.

We say that the language $L$ is in the class AM[poly] if there is a (polynomial-time) algorithm $V$ such that:

- When $x \in L$, there is always a strategy for $P$ to generate the responses to the random challenges that causes $V$ to accept.
- When $x \notin L$, regardless of how $P$ responds to the random challenges, with probability at least $1 - 2^{-|x|} = 1 - 2^{-n}$, $V$ rejects. Here the probability is taken over the random challenges.

In other words, we ask that there be a polynomial time algorithm $V$ such that if $x \in L$, there is always *some* response to the random challenge sequence that will convince $V$ of this fact; but if $x \notin L$, then *every* way of responding to the random challenge sequence has an overwhelming probability of being "caught" by $V$.

What is the power of the class AM[poly]? From the definition, it should be clear that every language in NP has an (easy) AM[poly] protocol in which $P$, the prover, ignores

the random challenges, and simply presents $V$ with the standard NP witness to $x \in L$ (e.g., a specific 3-coloring of the graph $G$). More surprisingly, every language in the class PSPACE (the class of all languages that can be recognized in deterministic polynomial space, conjectured to be much larger than NP) also has an AM[poly] protocol, a beautiful and important result due to [6, 9]. (For definitions of classes such as P, NP, and PSPACE, see [8, 4].)

If a language $L$ has an Arthur-Merlin game where Arthur asks only a *constant* number of questions, we say that $L \in$ AM[2]. NP corresponds to Arthur-Merlin games where Arthur says *nothing*, and thus clearly NP $\subseteq$ AM[2]. Restricting the number of questions seems to put severe limitations on the power of Arthur-Merlin games. Though AM[poly] = PSPACE, it is generally believed that

$$\text{NP} = \text{AM[2]} \subsetneq \text{PSPACE}.$$

## 4 DBN-MDPs Requiring Large Policies

In this section, we outline our construction proving that factored MDPs may not have any succinct representation for (even approximately) optimal policies, and conclude this note with a formal statement of the result.

Let us begin by drawing a high-level analogy with the MDP setting. Let $A$ be a language in PSPACE, and let $V$ and $P$ be the Turing machines for the AM[*poly*] protocol for $A$. Since $V$ is simply a Turing machine, it has some internal configuration $s$ (sufficient to completely describe the tape contents, read/write head position, abstract computational state, and so on) at any given moment in the protocol with $P$. Since we assume $P$ is all-powerful (computationally unbounded), we can assume that $P$ has complete knowledge of this internal state $s$ of $V$ at all times. The protocol at round $t$ can thus be viewed: $V$ is in some state/configuration $s_t$; a random bit sequence (the challenge) $b_t$ is generated; based on $s_t$ and $b_t$, $P$ computes some response or *action* $a_t$; and based on $b_t$ and $a_t$, $V$ enters its next configuration $s_{t+1}$. From this description, several observations can be made:

- $V$'s internal configuration $s_t$ constitutes state in the Markovian sense — combined with the action $a_t$, it entirely determines the next-state distribution. The dynamics are probabilistic due to the influence of the random bit sequence $b_t$.

- We can thus view $P$ as implementing a *policy* in the MDP determined by (the internal configuration of) $V$ — $P$'s actions, together with the stochastic $b_t$, determine the evolution of the $s_t$. Informally, we might imagine defining the total return to $P$ to be 1 if $P$ causes $V$ to accept, and 0 if $V$ rejects.

- The MDP so defined in this manner is not arbitrarily complex — in particular, the transition dynamics are defined by the polynomial-time Turing machine $V$.

At a high level, then, if every MDP so defined by a language in AM[poly] had an "efficient" policy $P$, then something remarkable would occur: the arbitrary power allowed to $P$ in the definition of the class would have been unnecessary. We shall see that this would have extraordinary and rather implausible complexity-theoretic implications. For the moment, let us simply sketch the refinements to this line of thought that will allow us to make the connection to factored MDPs: we will show that the MDPs defined above can actually be represented by DBN-MDPs with only constant indegree and a linear reward function. As suggested, this will allow us to assert rather strong negative results about even the *existence* of efficient policies, even when we ask for rather weak approximation to the optimal return.

We now turn to the problem of planning in a DBN-MDP. Typically, one might like to have a "general-purpose" planning procedure — a procedure that takes as input a description of a DBN-MDP $M = (Q, A, \delta, N, r)$, and returns a description of the optimal policy $\pi^*$.

This is what is typically meant by the term planning, and we note that it demands a certain kind of *uniformity* — a *single* planning algorithm that can efficiently compute a succinct representation of the optimal policy for any DBN-MDP. Note that the existence of such a planning algorithm would certainly imply that every DBN-MDP *has* a succinct representation of its optimal policy — but the converse does not hold. It could be that the difficulty of planning in DBN-MDPs arises from the demand of uniformity — that is, that every DBN-MDP *possesses* a succinct optimal policy, but the problem of *computing* it from the MDP parameters is intractable. This would be analogous to problems in NP — for example, every 3-colorable graph obviously has a succinct description of a 3-coloring, but it is difficult to compute it from the graph.

As mentioned in the introduction, it has been known for some time that planning in this uniform sense is computationally intractable. Here we establish the stronger and conceptually important result that it is *not* the uniformity giving rise to the difficulty, but rather that there simply exist DBN-MDPs in which the optimal policy does not possess a succinct representation in *any natural parameterization*. We will present a specific family of DBN-MDPs $\{M_n\}$ (where $M_n$ has states with $n$ components), and show that, under a standard complexity-theoretic assumption, the corresponding family of optimal policies $\{\pi_n^*\}$ cannot be represented by arbitrary Boolean circuits of size polynomial in $n$. We note that such circuits constitute a universal representation of efficiently computable functions, and all of the standard parametric forms in wide use in AI and statistics can be computed by such circuits.

We now provide the details of the construction. Let $A$ be any language in PSPACE, and let $V$ be a polynomial-time Turing machine running in time $m^k$ on inputs of length $m$, implementing the algorithm of "Arthur" in the AM protocol for $A$. Let $n$ be the maximum number of bits needed to write down a complete configuration of $V$ that may arise during computation on an input of length $m$ (so $n = O(m^k)$, since no computation taking $m^k$ time can consume more than $m^k$ space). Each state of our DBN-MDP $M_n$ will have $n$ components, each corresponding to one bit of the encoding of a configuration. No states will have rewards, except for the accepting states, which have reward $1/\gamma^{m^k}$. (Without loss of generality, we may assume that $V$ never enters an accepting state other than at time time $m^k$.) Note that we can encode configurations so that there is one bit position (say, the first bit of the state vector) that records if the current state of $V$ is accepting or not. Thus the reward function is obviously linear (it is simply $1/\gamma^{m^k}$ times the first component).

There are two actions: $A = \{0, 1\}$. Each action advances the simulation of the AM game by one time step. There are three types of steps:

1. Steps where $P$ is choosing a bit to send to $V$; action $b \in A$ corresponds to $P$ choosing to send a "$b$" to $V$.

2. Steps where $V$ is flipping a coin; each action $b \in A$ yields probability $1/2$ of having the coin come up "heads".

3. Steps where $V$ is doing deterministic computation; each action $b \in A$ moves the computation ahead one step.

It is straightforward to encode this as a DBN-MDP. Note that each bit of the next move relation of a Turing machine depends on only $O(1)$ bits of the preceding configuration (i.e., on the bits encoding the contents of the neighboring cells, the bits encoding the presence or absence of the input head in one of those cells, and the bits encoding the finite state information of the Turing machine). Thus the DBN-MDP $M_n$ describing $V$ on inputs of length $m$ has constant indegree; each bit is connected to the $O(1)$ bits on which it depends.

Note that a path in this MDP corresponding to an accepting computation of $V$ on an input of length $m$ has total reward 1; a rejecting path has reward 0. A routine calculation shows

that the expected reward of the optimal policy is equal to the fraction of coin flip sequences that cause $P$ to accept when communicating with an optimal $V$. That is,

$$\mathsf{Prob}(V \text{ accepts}) = (\text{Optimal expected reward})$$

With the construction above, we can now describe our result:

**Theorem 1.** *If PSPACE is not contained in* P/POLY, *then there is a family of DBN-MDPs* $M_n$, $n \geq 1$, *such that for any two polynomials,* $s(n)$ *and* $a(n)$, *there exist infinitely many* $n$ *such that no circuit* $C$ *of size* $s(n)$ *can compute a policy having expected reward greater than* $1/a(n)$ *times the optimum.*

Before giving the formal proof, we remark that the assumption that PSPACE is not contained in P/POLY is standard and widely believed, and informally asserts that not everything that can be computed in polynomial space can be computed by a non-uniform family of small circuits.

*Proof.* Let $A$ be any language in PSPACE that is not in P/POLY, and let $M_n$ be as described above. Suppose, contrary to the statement of Theorem, that for large enough $n$ there is indeed a circuit $C$ of size $s(n)$ computing a policy for $M_n$ whose return is within a $1/a(n)$ factor of optimal. We now consider the probabilistic circuit $D$ that operates as follows. $D$ takes a string $x$ as input, and estimates the expected return of the policy given by $C$ (which is the same as the probability that the prover $P$ associated with $C$ is able to convince $V$ that $x \in A$). Specifically, $D$ builds the state $q$ corresponding to the start state of $V$ protocol on input $x$, and then repeats the following procedure $a(n)$ times:

> Given state $q$, if $q$ is a state encoding a configuration in which it is $P$'s turn, use $C$ to compute the message sent by $P$ and set $q$ to the new state of the AM protocol.

> Otherwise, if $q$ is a state encoding a configuration in which it is $V$'s turn, flip a coin at random and set $q$ to the new state of the AM protocol. Repeat until an **accept** or **reject** state is encountered.

If any of these repetitions result in an **accept**, $D$ accepts; otherwise $D$ rejects.

Note now that if $x \in A$, then the probability that $D$ rejects is no more than

$$(1 - 1/a(n))^{a(n)} \approx 1/e \approx .37,$$

since in this case we are guaranteed that each iteration will accept with probability at least $1/a(n)$. On the other hand, if $x \notin A$, then $D$ accepts with probability no more than $a(n)/2^n$, since each iteration accepts with probability at most $2^{-n}$. As $D$ has polynomial size and a probabilistic circuit can be simulated by a deterministic one of essentially the same size, it follows that $A$ is in P/POLY, a contradiction. $\square$

It is worth mentioning that, by the worst-case-to-average-case reduction of [1], if PSPACE is not in P/POLY then we can select such a language $A$ so that the circuit $C$ will perform badly on a non-negligible fraction of the states $q$ of $M_n$. That is, not only is it hard to *find* an *optimal* policy, it will be the case that *every* policy that can be expressed as a polynomial size circuit will perform *very* badly on *very* many inputs.

Finally, we remark that by coupling the above construction with the approximate lower bound protocol of [3], one can prove (under a stronger assumption) that there are no succinct policies for the DBN-MDPs $M_n$ which even approximate the optimum return to within an *exponential* factor.

**Theorem 2.** *If* PSPACE *is not contained in* AM[2], *then there is a family of DBN-MDPs* $M_n, n \geq 1$, *such that for any polynomial $s$ there exist infinitely many $n$ such that no circuit $C$ of size $s(n)$ can compute a policy having expected reward greater than $2^{-n+1}$ times the optimum.*

## References

[1] L. Babai, L. Fortnow, N. Nisan, and A. Wigderson. BPP has subexponential time simulations unless EXPTIME has publishable proofs. *Complexity Theory*, 3:307–318, 1993.

[2] L. Babai and S. Moran. Arthur-merlin games: a randomized proof system, and a hierarchy of complexity classes. *Journal of Computer and System Sciences*, 36(2):254–276, 1988.

[3] S. Goldwasser and M. Sipser. Private coins versus public coins in interactive proof systems. *Advances in Computing Research*, 5:73–90, 1989.

[4] D. Johnson. A catalog of complexity classes. In J. van Leeuwen, editor, *Handbook of Theoretical Computer Science*, volume A. The MIT Press, 1990.

[5] P. Liberatore. The size of MDP factored policies. In *Proceedings of AAAI 2002*. AAAI Press, 2002.

[6] C. Lund, L. Fortnow, H. Karloff, and N. Nisan. Algebraic methods for interactive proof systems. *Journal of the ACM*, 39(4):859–868, 1992.

[7] M. Mundhenk, J. Goldsmith, C. Lusena, and E. Allender. Complexity of finite-horizon Markov decision process problems. *Journal of the ACM*, 47(4):681–720, 2000.

[8] C. Papadimitriou. *Computational Complexity*. Addison-Wesley, 1994.

[9] A. Shamir. IP = PSPACE. *Journal of the ACM*, 39(4):869–877, 1992.
